# One Microphone Source Separation

**Sam T. Roweis**
Gatsby Unit, University College London
roweis@gatsby.ucl.ac.uk

## Abstract

Source separation, or computational auditory scene analysis, attempts to extract individual acoustic objects from input which contains a mixture of sounds from different sources, altered by the acoustic environment. Unmixing algorithms such as ICA and its extensions recover sources by reweighting multiple observation sequences, and thus cannot operate when only a single observation signal is available. I present a technique called *refiltering* which recovers sources by a nonstationary reweighting ("masking") of frequency sub-bands from a single recording, and argue for the application of statistical algorithms to learning this masking function. I present results of a simple factorial HMM system which learns on recordings of single speakers and can then separate mixtures using only one observation signal by computing the masking function and then refiltering.

## 1 Learning from data in computational auditory scene analysis

Imagine listening to many pianos being played simultaneously. If each pianist were striking keys randomly it would be very difficult to tell which note came from which piano. But if each were playing a coherent song, separation would be much easier because of the structure of music. Now imagine teaching a computer to do the separation by showing it many musical scores as "training data". Typical auditory perceptual input contains a mixture of sounds from different sources, altered by the acoustic environment. Any biological or artificial hearing system must extract individual acoustic objects or *streams* in order to do successful localization, denoising and recognition. Bregman [1] called this process *auditory scene analysis* in analogy to vision. Source separation, or *computational auditory scene analysis* (CASA) is the practical realization of this problem via computer analysis of microphone recordings and is very similar to the musical task described above. It has been investigated by research groups with different emphases. The CASA community have focused on both multiple and single microphone source separation problems under highly realistic acoustic conditions, but have used almost exclusively hand designed systems which include substantial knowledge of the human auditory system and its psychophysical characteristics (e.g. [2,3]). Unfortunately, it is difficult to incorporate large amounts of detailed statistical knowledge about the problem into such an approach. On the other hand, machine learning researchers, especially those working on independent components analysis (ICA) and related algorithms, have focused on the case of multiple microphones in simplified mixing environments and have used powerful "blind" statistical techniques. These "unmixing" algorithms (even those which attempt to recover more sources than signals) cannot operate on single recordings. Furthermore, since they often depend only on the joint amplitude histogram of the observations they can be very sensitive to the details of filtering and reverberation in the environment. The goal of this paper is to bring together the robust representations of CASA and methods which learn from data to solve a restricted version of the source separation problem – isolating acoustic objects from only a single microphone recording.

## 2 Refiltering vs. unmixing

*Unmixing* algorithms reweight multiple simultaneous recordings $m_k(t)$ (generically called microphones) to form a new source object $s(t)$:

$$\underbrace{s(t)}_{\text{estimated source}} = \alpha_1 \underbrace{m_1(t)}_{\text{mic 1}} + \alpha_2 \underbrace{m_2(t)}_{\text{mic 2}} + \ldots + \alpha_K \underbrace{m_K(t)}_{\text{mic K}} \qquad (1)$$

The unmixing coefficients $\alpha_i$ are constant over time and are chosen to optimize some property of the set of recovered sources, which often translates into a kurtosis measure on the joint amplitude histogram of the microphones. The intuition is that unmixing algorithms are finding spikes (or dents for low kurtosis sources) in the marginal amplitude histogram. The time ordering of the datapoints is often irrelevant.

Unmixing depends on a fine timescale, sample-by-sample comparison of several observation signals. Humans, on the other hand, cannot hear histogram spikes[1] and perform well on many monaural separation tasks. We are doing structural analysis, or a kind of perceptual grouping on the incoming sound. But what is being grouped? There is substantial evidence that the energy across time in different frequency bands can carry relatively independent information. This suggests that the appropriate subparts of an audio signal may be narrow frequency bands over short times. To generate these parts, one can perform multiband analysis – break the original signal $y(t)$ into many subband signals $b_i(t)$ each filtered to contain only energy from a small portion of the spectrum. The results of such an analysis are often displayed as a *spectrogram* which shows energy (using colour or grayscale) as a function of time (ordinate) and frequency (abscissa). (For example one is shown on the top left of figure 5.) In the musical analogy, a spectrogram is like a musical score in which the colour or grey level of the each note tells you how hard to hit the piano key.

The basic idea of *refiltering* is to construct new sources by selectively reweighting the multiband signals $b_i(t)$. Crucially, however, the mixing coefficients are no longer constant over time; they are now called *masking signals*. Given a set of masking signals, denoted $\alpha_i(t)$, a source $s(t)$ can be recovered by modulating the corresponding subband signals from the original input and summing:

$$\underbrace{s(t)}_{\text{estimated source}} = \overbrace{\alpha_1(t)}^{\text{mask 1}} \underbrace{b_1(t)}_{\text{sub-band 1}} + \overbrace{\alpha_2(t)}^{\text{mask 2}} \underbrace{b_2(t)}_{\text{sub-band 2}} + \ldots + \overbrace{\alpha_K(t)}^{\text{mask K}} \underbrace{b_K(t)}_{\text{sub-band K}} \qquad (2)$$

The $\alpha_i(t)$ are gain knobs on each subband that we can twist over time to bring bands in and out of the source as needed. This performs masking on the original spectrogram. (An equivalent operation can be performed in the frequency domain.[2]) This approach, illustrated in figure 1, forms the basis of many CASA approaches (e.g. [2,3,4]).

For any specific choice of masking signals $\alpha_i(t)$, refiltering attempts to isolate a *single source* from the input signal and suppress all other sources and background noises. Different sources can be isolated by choosing different masking signals. Henceforth, I will make a strong simplifying assumption that $\alpha_i(t)$ are binary and constant over a timescale $\tau$ of roughly 30ms. This is physically unrealistic, because the energy in each small region of time-frequency never comes entirely from a single source. However in practice, for small numbers of sources, this approximation works quite well (figure 3). (Think of ignoring collisions by assuming separate piano players do not often hit the same note at the same time.)

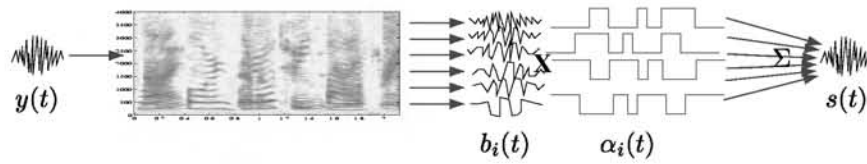

$y(t)$          $b_i(t)$    $\alpha_i(t)$         $s(t)$

**Figure 1:** The refiltering approach to one microphone source separation. Multiband analysis of the original signal $y(t)$ gives sub-band signals $b_i(t)$ which are modulated by masking signals $\alpha_i(t)$ (binary or real valued between 0 and 1) and recombined to give the estimated source or object $s(t)$.

Refiltering can also be thought of as a highly nonstationary Wiener filter in which both the signal and noise spectra are re-estimated at a rate $1/\tau$; the binary assumption is equivalent to assuming that over a timescale $\tau$ the signal and noise spectra are nonoverlapping.

It is a fortunate empirical fact that refiltering, even with binary masking signals, *can* cleanly separate sources from a single mixed recording. This can be demonstrated by taking several isolated sources or noises and mixing them in a controlled way. Since the original components are known, an "optimal" set of masking signals can be computed. For example, we might set $\alpha_i(t)$ equal to the ratio of energy from one source in band $i$ around times $t \pm \tau$ to the sum of energies from all sources in the same band at that time (as recommended by the Wiener filter) or to a binary version which thresholds this ratio. Constructing masks in this way is also useful for generating labeled training data, as discussed below.

## 3 Multiband grouping as a statistical pattern recognition problem

Since one-microphone source separation using refiltering is possible if the masking signals are well chosen, the essential problem becomes: how can the $\alpha_i(t)$ be computed automatically from a single mixed recording? The goal is to group or "tag" together regions of the spectrogram that belong to the same auditory object. Fortunately, in audition (as in vision), natural signals—especially speech—exhibit a lot of regularity in the way energy is distributed across the time-frequency plane. Grouping cues based on these regularities have been studied for many years by psychophysicists and are hand built into many CASA systems. Cues are based on the idea of suspicious coincidences: roughly, "things that move together likely belong together". Thus, frequencies which exhibit common onsets, offsets, or upward/downward sweeps are more likely to be grouped into the same stream (figure 2). Also, many real world sounds have harmonic spectra; so frequencies which lie exactly on a harmonic "stack" are often perceptually grouped together. (Musically, piano players do not hit keys randomly, but instead use chords and repeated melodies.)

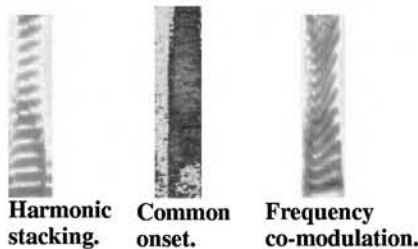

**Harmonic stacking.**    **Common onset.**    **Frequency co-modulation.**

**Figure 2:** Examples of three common grouping cues for energy which often comes from a single source. **(left)** Frequencies which lie exactly on harmonic multiples of a single base frequency. **(middle)** Frequencies which suddenly increase or decrease their energy together. **(right)** Energy which which moves up or down in frequency at the same time.

There are several ways that statistical pattern recognition might be applied to take advantage of these cues. Methods may be roughly grouped into unsupervised ones, which learn models of isolated sources and then try to explain mixed input as being caused by the interaction of individual source models; and supervised methods, which explicitly model grouping in mixed acoustic input but require labeled data consisting of mixed input as well

as masking signals. Luckily it is very easy to generate such data by mixing isolated sources in a controlled way, although the subsequent supervised learning can difficult.[3]

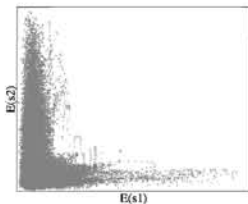

**Figure 3:** Each point represents the energy from one source versus another in a narrow frequency band over a 32ms window. The plot shows all frequencies over a 2 second period from a speech mixture. Typically when one source has large energy the other does not. The binary assumption on the masking signals $\alpha_i(t)$ is equivalent to projecting the points shown onto either the horizontal or vertical axis.

## 4  Results using factorial-max HMMs

Here, I will describe one (purely unsupervised) method I have pursued for automatically generating masking signals from a single microphone. The approach first trains speaker dependent hidden Markov models (HMMs) on isolated data from single talkers. These pre-trained models are then combined in a particular way to build a separation system.

First, for each speaker, a simple HMM is fit using patches of narrowband spectrograms as the pattern vectors.[4] The emission densities model the typical spectral patterns produced by each talker, while the transition probabilities encourage spectral continuity. HMM training was initialized by first training a mixture of Gaussians on each speaker's data (with a single shared covariance matrix) independent of time order. Each mixture had 8192 components of dimension $1026 = 513 \times 2$; thus each HMM had 8192 states. To avoid overfitting, the transition matrices were regularlized after training so that each transition (even those unobserved in the training set) had a small finite probability.

Next, to separate a new single recording which is a mixture of known speakers, these pre-trained models are combined into a *factorial hidden Markov model* (FHMM) architecture [5]. A FHMM consists of two or more underlying Markov chains (the hidden states) which evolve independently. The observation $\mathbf{y}_t$ at any time depends on the states of all the chains. A simple way to model this dependence is to have each chain $c$ independently propose an output $\mathbf{y}_t^c$ and then combine them to generate the observation according to some rule $\mathbf{y}_t = Q(\mathbf{y}_t^1, \mathbf{y}_t^2, \ldots, \mathbf{y}_t^c)$. Below, I use a model with only two chains, whose states are denoted $x_t$ and $z_t$. At each time, one chain proposes an output vector $\mathbf{a}_{x_t}$ and the other proposes $\mathbf{b}_{z_t}$. The key part of the model is the function $Q$: observations are generated by taking the *elementwise maximum* of the proposals and adding noise. This maximum operation reflects the observation that the log magnitude spectrogram of a mixture of sources is very nearly the elementwise maximum of the individual spectrograms. The full generative model for this "factorial-max HMM" can be written simply as:

$$p(x_t = j | x_{t-1} = i) = T_{ij} \tag{3}$$

$$p(z_t = j | z_{t-1} = i) = U_{ij} \tag{4}$$

$$p(\mathbf{y}_t | x_t, z_t) = \mathcal{N}(\max[\mathbf{a}_{x_t}, \mathbf{b}_{z_t}], R) \tag{5}$$

where $\mathcal{N}(\mu, \Sigma)$ denotes a Gaussian distribution with mean $\mu$ and covariance $\Sigma$ and $\max[\cdot]$ is the elementwise maximum operation on two vectors. (There are also densities on the initial states $x_1$ and $z_1$.) This model is illustrated in figure 4. It ignores two aspects of the spectrogram data: first, Gaussian noise is used although the observations are nonnegative; second, the probability factor requiring the non-maximum output proposal to be less than the maximum proposal is missing. However, in practice these approximations are not too severe and making them allows an efficient inference procedure (see below).

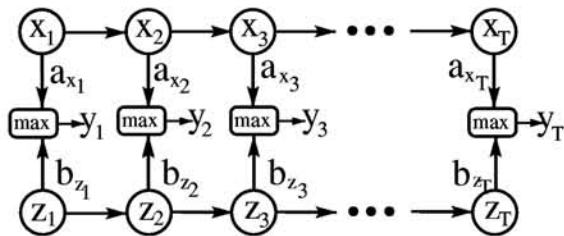

**Figure 4:** Factorial HMM with max output semantics. Two Markov chains $x_t$ and $z_t$ evolve independently. Observations $\mathbf{y}_t$ are the elementwise max of the individual emission vectors $\max[\mathbf{a}_{x_t}, \mathbf{b}_{z_t}]$ plus Gaussian noise.

In the experiment presented below, each chain represents a speaker dependent HMM (one male and one female). The emission and transition probabilities from each speaker's pre-trained HMM were used as the parameters for the combined FHMM. (The output noise covariance $R$ is shared between the two HMMs.)

Given an input waveform, the observation sequence $Y = \mathbf{y}_1, \ldots, \mathbf{y}_T$ is created from the spectrogram as before.[4] Separation is done by first inferring a joint underlying state sequence $\{\hat{x}_t, \hat{z}_t\}$ of the two Markov chains in the model and then using the difference of their individual output predictions to compute a binary masking signal:

$$\alpha_t(i) = 1 \quad \text{if} \quad \mathbf{a}_{\hat{x}_t}(i) > \mathbf{b}_{\hat{z}_t}(i) \quad \text{and} \quad 0 \quad \text{if} \quad \mathbf{a}_{\hat{x}_t}(i) \leq \mathbf{b}_{\hat{z}_t}(i) \tag{6}$$

Ideally, the inferred state sequences $\{\hat{x}_t, \hat{z}_t\}$ should be the mode of the posterior distribution $p(x_t, z_t | Y)$. Since the hidden chains share a single visible output variable, naive inference in the FHMM graphical model yields an intractable amount of work exponential in the size of the state space of each submodel. However, because all of the observations are nonnegative and the `max` operation is used to combine output proposals, there is an efficient trick for computing the best joint state trajectory. At each time, we can *upper bound* the log-probability of generating the observation vector if one chain is in state $i$, *no matter what state the other chain is in*. Computing these bounds for each state setting of each chain requires only a linear amount of work in the size of the state spaces. With these bounds in hand, each time we evaluate the probability of a specific *pair* of states we can eliminate from consideration all state settings of either chain whose bounds are worse than the achieved probability. If pairs of states are evaluated in a sensible heuristic order (for example by ranking the bounds) this results in practice in almost all possible configurations being quickly eliminated. (This trick turns out to be equivalent to $\alpha\beta$ search in game trees.)

The training data for the model consists only of spectrograms of isolated examples of each speaker but inference can be done on test data which is a spectrogram of a single mixture of known speakers. The results of separating a simple two speaker mixture are shown below. The test utterance was formed by linearly mixing two out-of-sample utterances (one male and one female) from the same speakers as the models were trained on. Figure 5 shows the original mixed spectrogram (top left) as well as the sequence of outputs $\mathbf{a}_{\hat{x}_t}$ (bottom left) and $\mathbf{b}_{\hat{z}_t}$ (bottom right) from each chain. The chain with the maximum output in any sub-band at any time has $\alpha_i(t) = 1$, otherwise $\alpha_i(t) = 0$ (top right). The FHMM system achieves good separation from only a single microphone (see figure 6).

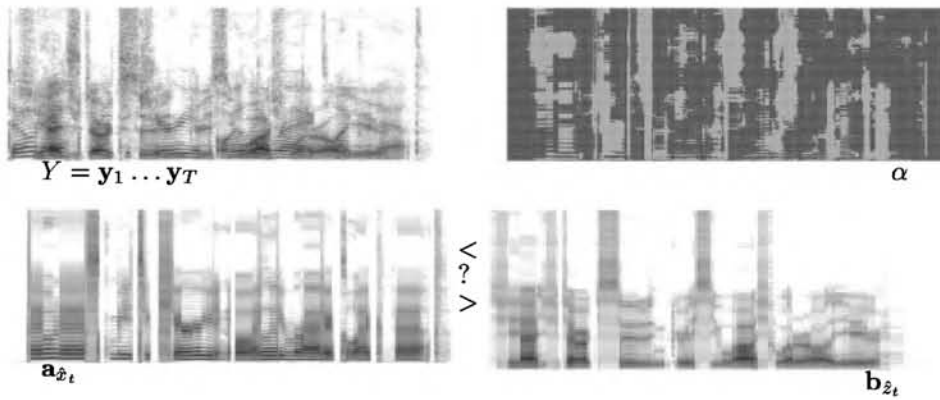

$Y = \mathbf{y}_1 \ldots \mathbf{y}_T$      $\alpha$

$\mathbf{a}_{\hat{x}_t}$      $<?>$      $\mathbf{b}_{\hat{z}_t}$

**Figure 5:** **(top left)** Original spectrogram of mixed utterance. **(bottom)** Male and female spectrograms predicted by factorial HMM and used to compute refiltering masks. **(top right)** Masking signals $\alpha_i(t)$, computed by comparing the magnitudes of each model's predictions.

## 5 Conclusions

In this paper I have argued for the marriage of learning algorithms with the refiltering approach to CASA. I have presented results from a simple factorial HMM system on a speaker dependent separation problem which indicate that automatically learned one-microphone separation systems may be possible. In the machine learning community, the one-microphone separation problem has received much less attention than unmixing problems, while CASA researchers have not employed automatic learning techniques to full effect. Scene analysis is an interesting and challenging learning problem with exciting and practical applications, and the refiltering setup has many nice properties. First, it *can* work if the masking signals are chosen properly. Second, it is easy to generate lots of training data, both supervised and unsupervised. Third, a good learning algorithm—when presented with enough data—should automatically discover the sorts of grouping cues which have been built into existing systems by hand.

Furthermore, in the refiltering paradigm there is no need to make a hard decision about the number of sources present in an input. Each proposed masking has an associated score or probability; groupings with high scores can be considered "sources", while ones with low scores might be parts of the background or mixtures other faint sources. CASA returns a collection of candidate maskings and their associated scores, and then it is up to the user to decide—based on the range of scores—the number of sources in the scene.

Many existing approaches to speech and audio processing have the potential to be applied to the monaural source separation problem. The unsupervised factorial HMM system presented in this paper is very similar to the work in the speech recognition community on *parallel model combination* [6,7]; however rather than using the combined models to evaluate the likelihood of speech in noise, the efficiently inferred states are being used to generate a masking signal for refiltering. Wan and Nelson have developed dual EKF methods [8] and applied them speech denoising but have also informally demonstrated their potential application to monaural source separation. Attias and colleagues [9] developed a fully probabilistic model of speech in noise and used variational Bayesian techniques to perform inference and learning allowing denoising and dereverberation; their approach clearly has the potential to be applied to the separation problem as well. Cauwenberghs [10] has a very promising approach to the problem for purely harmonic signals that takes advantage of powerful phase constraints which are ignored by other algorithms.

Unsupervised and supervised approaches can be combined to various degrees. Learning models of isolated sounds may be useful for developing feature detectors; conjunctions of such feature detectors can then be trained in a supervised fashion using labeled data.

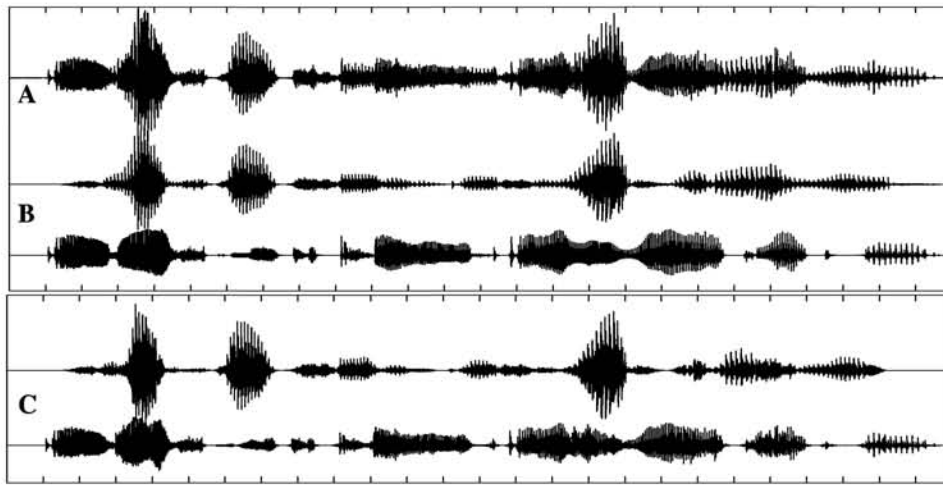

**Figure 6:** Test separation results, using a 2-chain speaker dependent factorial-max HMM, followed by refiltering. (See figure 4 and text for details.) **(A)** Original waveform of mixed utterance. **(B)** Original isolated male & female waveforms. **(C)** Estimated male and female waveforms.

The oscillatory correlation algorithm of Brown and Wang [4] has a low level module to detect features in the correlogram and a high level module to do grouping. Related ideas in machine vision, such as Markov networks [11] and minimum normalized cut [12] use low level operations to define weights between pixels and then higher level computations to group pixels together.

### Acknowledgements

Thanks to Hagai Attias, Guy Brown, Geoff Hinton and Lawrence Saul for many insightful discussions about the CASA problem, and to three anonymous referees and many visitors to my poster for helpful comments, criticisms and references to work I had overlooked.

## Footnotes

[1] Try randomly permuting the time order of samples in a stereo mixture containing several sources and see if you still hear distinct streams when you play it back.

[2] Make a conventional spectrogram of the original signal $y(t)$ and modulate the magnitude of each short time DFT while preserving its phase: $s^w(\tau) = \mathcal{F}^{-1}\{\alpha^w\|\mathcal{F}\{y^w(\tau)\}\|\angle\mathcal{F}\{y^w(\tau)\}\}$ where $s^w(\tau)$ and $y^w(\tau)$ are the $w^{th}$ windows (blocks) of the recovered and original signals, $\alpha_i^w$ is the masking signal for subband $i$ in window $w$, and $\mathcal{F}[\cdot]$ is the DFT.

[3]Recall that refiltering can only isolate one auditory stream at a time from the scene (we are always separating "a source" from "the background"). This makes learning the masking signals an unusual problem because for any input (spectrogram) there are as many correct answers as objects in the scene. Such a highly multimodal distribution on outputs given inputs means that the mapping from auditory input to masking signals cannot be learned using backprop or other single-valued function approximators which take the *average* of the possible maskings present in the training data.

[4]The observations are created by concatenating the values of 2 adjacent columns of the log magnitude periodogram into a single vector. The original waveforms were sampled at 16kHz. Periodogram windows of 32ms at a frame rate of 16ms were analyzed using a Hamming tapered DFT zero padded to length 1024. This gave 513 frequency samples from DC to Nyquist. Average signal energy was normalized across the most recent 8 frames before computing each DFT.

### References

[1] A.S. Bregman. (1994) *Auditory Scene Analysis.* MIT Press.
[2] G. Brown & M. Cooke. (1994) *Computational auditory scene analysis.* Computer Speech and Language 8.
[3] D. Ellis. (1994) *A computer implementation of psychoacoustic grouping rules.* Proc. 12th Intl. Conf. on Pattern Recognition, Jerusalem.
[4] G. Brown & D.L. Wang. (2000) *An oscillatory correlation framework for computational auditory scene analysis.* NIPS 12.
[5] Z. Ghahramani & M.I. Jordan (1997) *Factorial hidden Markov models*, Machine Learning 29.
[6] A.P. Varga & R.K. Moore (1990) *Hidden Markov model decomposition of speech and noise*, IEEE Conf. Acoustics, Speech & Signal Processing (ICASSP'90).
[7] M.J.F. Gales & S.J. Young (1996) *Robust continuous speech recognition using parallel model combination*, IEEE Trans. Speech & Audio Processing 4.
[8] E.A. Wan & A.T. Nelson (1998) *Removal of noise from speech using the dual EKF algorithm*, IEEE Conf. Acoustics, Speech & Signal Processing (ICASSP'98).
[9] H. Attias, J.C. Platt & A. Acero (2001) *Speech denoising and dereverberation using probabilistic models*, this volume.
[10] G. Cauwenberghs (1999) *Monaural separation of independent acoustical components*, IEEE Symp. Circuit & Systems (ISCAS'99).
[11] W. Freeman & E. Pasztor. (1999) *Markov networks for low-level vision.* Mitsubishi Electric Research Laboratory Technical Report TR99-08.
[12] J. Shi & J. Malik. (1997) *Normalized cuts and image segmentation.* IEEE Conf. Computer Vision and Pattern Recognition, Puerto Rico (ICCV'97).
